# Efficient and Robust Feature Selection via Joint $\ell_{2,1}$-Norms Minimization

**Feiping Nie**
Computer Science and Engineering
University of Texas at Arlington
feipingnie@gmail.com

**Heng Huang**
Computer Science and Engineering
University of Texas at Arlington
heng@uta.edu

**Xiao Cai**
Computer Science and Engineering
University of Texas at Arlington
xiao.cai@mavs.uta.edu

**Chris Ding**
Computer Science and Engineering
University of Texas at Arlington
chqding@uta.edu

## Abstract

Feature selection is an important component of many machine learning applications. Especially in many bioinformatics tasks, efficient and robust feature selection methods are desired to extract meaningful features and eliminate noisy ones. In this paper, we propose a new robust feature selection method with emphasizing joint $\ell_{2,1}$-norm minimization on both loss function and regularization. The $\ell_{2,1}$-norm based loss function is robust to outliers in data points and the $\ell_{2,1}$-norm regularization selects features across all data points with joint sparsity. An efficient algorithm is introduced with proved convergence. Our regression based objective makes the feature selection process more efficient. Our method has been applied into both genomic and proteomic biomarkers discovery. Extensive empirical studies are performed on six data sets to demonstrate the performance of our feature selection method.

## 1 Introduction

Feature selection, the process of selecting a subset of relevant features, is a key component in building robust machine learning models for classification, clustering, and other tasks. Feature section has been playing an important role in many applications since it can speed up the learning process, improve the mode generalization capability, and alleviate the effect of the *curse of dimensionality* [15]. A large number of developments on feature selection have been made in the literature and there are many recent reviews and workshops devoted to this topic, *e.g.*, NIPS Conference [7].

In past ten years, feature selection has seen much activities primarily due to the advances in bioinformatics where a large amount of genomic and proteomic data are produced for biological and biomedical studies. For example, in genomics, DNA microarray data measure the expression levels of thousands of genes in a single experiment. Gene expression data usually contain a large number of genes, but a small number of samples. A given disease or a biological function is usually associated with a few genes [19]. Out of several thousands of genes to select a few of relevant genes thus becomes a key problem in bioinformatics research [22]. In proteomics, high-throughput mass spectrometry (MS) screening measures the molecular weights of individual biomolecules (such as proteins and nucleic acids) and has potential to discover putative proteomic biomarkers. Each spectrum is composed of peak amplitude measurements at approximately 15,500 features represented by a corresponding mass-to-charge value. The identification of meaningful proteomic features from MS is crucial for disease diagnosis and protein-based biomarker profiling [22].

In general, there are three models of feature selection methods in the literature: (1) filter methods [14] where the selection is independent of classifiers, (2) wrapper methods [12] where the prediction method is used as a black box to score subsets of features, and (3) embedded methods where the procedure of feature selection is embedded directly in the training process. In bioinformatics applications, many feature selection methods from these categories have been proposed and applied. Widely used filter-type feature selection methods include $F$-statistic [4], reliefF [11, 13], mRMR [19], t-test, and information gain [21] which compute the sensitivity (correlation or relevance) of a feature with respect to (w.r.t) the class label distribution of the data. These methods can be characterized by using global statistical information. Wrapper-type feature selection methods is tightly coupled with a specific classifier, such as correlation-based feature selection (CFS) [9], support vector machine recursive feature elimination (SVM-RFE) [8]. They often have good performance, but their computational cost is very expensive.

Recently sparsity regularization in dimensionality reduction has been widely investigated and also applied into feature selection studies. $\ell_1$-SVM was proposed to perform feature selection using the $\ell_1$-norm regularization that tends to give sparse solution [3]. Because the number of selected features using $\ell_1$-SVM is upper bounded by the sample size, a Hybrid Huberized SVM (HHSVM) was proposed combining both $\ell_1$-norm and $\ell_2$-norm to form a more structured regularization [26]. But it was designed only for binary classification. In multi-task learning, in parallel works, Obozinsky *et. al.* [18] and Argyriou *et. al.* [1] have developed a similar model for $\ell_{2,1}$-norm regularization to couple feature selection across tasks. Such regularization has close connections to group lasso [28].

In this paper, we propose a novel efficient and robust feature selection method to employ joint $\ell_{2,1}$-norm minimization on both loss function and regularization. Instead of using $\ell_2$-norm based loss function that is sensitive to outliers, a $\ell_{2,1}$-norm based loss function is adopted in our work to remove outliers. Motivated by previous research [1, 18], a $\ell_{2,1}$-norm regularization is performed to select features across all data points with joint sparsity, *i.e.* each feature (gene expression or mass-to-charge value in MS) either has small scores for all data points or has large scores over all data points. To solve this new robust feature selection objective, we propose an efficient algorithm to solve such joint $\ell_{2,1}$-norm minimization problem. We also provide the algorithm analysis and prove the convergence of our algorithm. Extensive experiments have been performed on six bioinformatics data sets and our method outperforms five other commonly used feature selection methods in statistical learning and bioinformatics.

## 2   Notations and Definitions

We summarize the notations and the definition of norms used in this paper. Matrices are written as boldface uppercase letters. Vectors are written as boldface lowercase letters. For matrix $\mathbf{M} = (m_{ij})$, its $i$-th row, $j$-th column are denoted by $\mathbf{m}^i$, $\mathbf{m}_j$ respectively.

The $\ell_p$-norm of the vector $\mathbf{v} \in \mathbb{R}^n$ is defined as $\|\mathbf{v}\|_p = \left( \sum_{i=1}^{n} |v_i|^p \right)^{\frac{1}{p}}$. The $\ell_0$-norm of the vector $\mathbf{v} \in \mathbb{R}^n$ is defined as $\|\mathbf{v}\|_0 = \sum_{i=1}^{n} |v_i|^0$. The Frobenius norm of the matrix $\mathbf{M} \in \mathbb{R}^{n \times m}$ is defined as

$$\|\mathbf{M}\|_F = \sqrt{\sum_{i=1}^{n} \sum_{j=1}^{m} m_{ij}^2} = \sqrt{\sum_{i=1}^{n} \|\mathbf{m}^i\|_2^2}. \tag{1}$$

The $\ell_{2,1}$-norm of a matrix was first introduced in [5] as rotational invariant $\ell_1$ norm and also used for multi-task learning [1, 18] and tensor factorization [10]. It is defined as

$$\|\mathbf{M}\|_{2,1} = \sum_{i=1}^{n} \sqrt{\sum_{j=1}^{m} m_{ij}^2} = \sum_{i=1}^{n} \|\mathbf{m}^i\|_2, \tag{2}$$

which is rotational invariant for rows: $\|\mathbf{MR}\|_{2,1} = \|\mathbf{M}\|_{2,1}$ for any rotational matrix $\mathbf{R}$. The $\ell_{2,1}$-norm can be generalized to $\ell_{r,p}$-norm

$$\|\mathbf{M}\|_{r,p} = \left( \sum_{i=1}^{n} \left( \sum_{j=1}^{m} |m_{ij}|^r \right)^{\frac{p}{r}} \right)^{\frac{1}{p}} = \left( \sum_{i=1}^{n} \|\mathbf{m}^i\|_r^p \right)^{\frac{1}{p}}. \tag{3}$$

Note that $\ell_{r,p}$-norm is a valid *norm* because it satisfies the three norm conditions, including the triangle inequality $\|\mathbf{A}\|_{r,p} + \|\mathbf{B}\|_{r,p} \geq \|\mathbf{A} + \mathbf{B}\|_{r,p}$. This can be proved as follows. Starting from the triangle inequality $(\sum_i |u_i|^p)^{\frac{1}{p}} + (\sum_i |v_i|^p)^{\frac{1}{p}} \geq (\sum_i |u_i + v_i|^p)^{\frac{1}{p}}$ and setting $u_i = \|\mathbf{a}^i\|_r$ and $v_i = \|\mathbf{b}^i\|_r$, we obtain

$$\left( \sum_i \|\mathbf{a}^i\|_r^p \right)^{\frac{1}{p}} + \left( \sum_i \|\mathbf{b}^i\|_r^p \right)^{\frac{1}{p}} \geq \left( \sum_i | \|\mathbf{a}^i\|_r + \|\mathbf{b}^i\|_r|^p \right)^{\frac{1}{p}} \geq \left( \sum_i | \|\mathbf{a}^i + \mathbf{b}^i\|_r|^p \right)^{\frac{1}{p}}, \tag{4}$$

where the second inequality follows the triangle inequality for $\ell_r$ norm: $\|\mathbf{a}^i\|_r + \|\mathbf{b}^i\|_r \geq \|\mathbf{a}^i + \mathbf{b}^i\|_r$. Eq. (4) is just $\|\mathbf{A}\|_{r,p} + \|\mathbf{B}\|_{r,p} \geq \|\mathbf{A} + \mathbf{B}\|_{r,p}$.

However, the $\ell_0$-norm is not a valid norm because it does not satisfy the positive scalability: $\|\alpha \mathbf{v}\|_0 = |\alpha| \|\mathbf{v}\|_0$ for scalar $\alpha$. The term "*norm*" here is for convenience.

# 3 Robust Feature Selection Based on $\ell_{2,1}$-Norms

Least square regression is one of the popular methods for classification. Given training data $\{\mathbf{x}_1, \mathbf{x}_2, \cdots, \mathbf{x}_n\} \in \mathbb{R}^d$ and the associated class labels $\{\mathbf{y}_1, \mathbf{y}_2, \cdots, \mathbf{y}_n\} \in \mathbb{R}^c$, traditional least square regression solves the following optimization problem to obtain the projection matrix $\mathbf{W} \in \mathbb{R}^{d \times c}$ and the bias $\mathbf{b} \in \mathbb{R}^c$:

$$\min_{\mathbf{W},\mathbf{b}} \sum_{i=1}^{n} \left\| \mathbf{W}^T \mathbf{x}_i + \mathbf{b} - \mathbf{y}_i \right\|_2^2. \tag{5}$$

For simplicity, the bias $\mathbf{b}$ can be absorbed into $\mathbf{W}$ when the constant value 1 is added as an additional dimension for each data $x_i (1 \leq i \leq n)$. Thus the problem becomes:

$$\min_{\mathbf{W}} \sum_{i=1}^{n} \left\| \mathbf{W}^T \mathbf{x}_i - \mathbf{y}_i \right\|_2^2. \tag{6}$$

In this paper, we use the robust loss function:

$$\min_{\mathbf{W}} \sum_{i=1}^{n} \left\| \mathbf{W}^T \mathbf{x}_i - \mathbf{y}_i \right\|_2, \tag{7}$$

where the residual $\|\mathbf{W}^T \mathbf{x}_i - \mathbf{y}_i\|$ is not squared and thus outliers have less importance than the squared residual $\|\mathbf{W}^T \mathbf{x}_i - \mathbf{y}_i\|^2$. This loss function has a rotational invariant property while the pure $\ell_1$-norm loss function does not has such desirable property [5].

We now add a regularization term $R(\mathbf{W})$ with parameter $\gamma$. The problem becomes:

$$\min_{\mathbf{W}} \sum_{i=1}^{n} \left\| \mathbf{W}^T \mathbf{x}_i - \mathbf{y}_i \right\|_2 + \gamma R(\mathbf{W}). \tag{8}$$

Several regularizations are possible:

$$R_1(\mathbf{W}) = \|\mathbf{W}\|^2, \ R_2(\mathbf{W}) = \sum_{j=1}^{c} \|\mathbf{w}_j\|_1, \ R_3(\mathbf{W}) = \sum_{i=1}^{d} \|\mathbf{w}^i\|_2^0, \ R_4(\mathbf{W}) = \sum_{i=1}^{d} \|\mathbf{w}^i\|_2. \tag{9}$$

$R_1(\mathbf{W})$ is the ridge regularization. $R_2(\mathbf{W})$ is the LASSO regularization. $R_3(\mathbf{W})$ and $R_4(\mathbf{W})$ penalizes all $c$ regression coefficients corresponding to a single feature as a whole. This has the

effects of feature selection. Although the $\ell_0$-norm of $R_3(\mathbf{W})$ is the most desirable [16], in this paper, we use $R_4(\mathbf{W})$ instead. The reasons are: (A) the $\ell_1$-norm of $R_4(\mathbf{W})$ is convex and can be easily optimized (the main contribution of this paper); (B) it was shown that results of $\ell_0$-norm is identical or approximately identical to the $\ell_1$-norm results under practical conditions.

Denote data matrix $\mathbf{X} = [\mathbf{x}_1, \mathbf{x}_2, \cdots, \mathbf{x}_n] \in \mathbb{R}^{d \times n}$ and label matrix $\mathbf{Y} = [\mathbf{y}_1, \mathbf{y}_2, \cdots, \mathbf{y}_n]^T \in \mathbb{R}^{n \times c}$. In this paper, we optimize

$$\min_{\mathbf{W}} \ J(\mathbf{W}) = \sum_{i=1}^{n} \left\| \mathbf{W}^T \mathbf{x}_i - \mathbf{y}_i \right\|_2 + \gamma R_4(\mathbf{W}) = \left\| \mathbf{X}^T \mathbf{W} - \mathbf{Y} \right\|_{2,1} + \gamma \left\| \mathbf{W} \right\|_{2,1}. \qquad (10)$$

It seems that solving this joint $\ell_{2,1}$-norm problem is difficult as both of the terms are non-smooth. Surprisingly, we will show in the next section that the problem can be solved using a simple yet efficient algorithm.

## 4 An Efficient Algorithm

### 4.1 Reformulation as A Constrained Problem

First, the problem in Eq. (10) is equivalent to

$$\min_{\mathbf{W}} \frac{1}{\gamma} \left\| \mathbf{X}^T \mathbf{W} - \mathbf{Y} \right\|_{2,1} + \left\| \mathbf{W} \right\|_{2,1}, \qquad (11)$$

which is further equivalent to

$$\min_{\mathbf{W}, \mathbf{E}} \left\| \mathbf{E} \right\|_{2,1} + \left\| \mathbf{W} \right\|_{2,1} \qquad \text{s.t.} \qquad \mathbf{X}^T \mathbf{W} + \gamma \mathbf{E} = \mathbf{Y}. \qquad (12)$$

Rewriting the above problem as

$$\min_{\mathbf{W}, \mathbf{E}} \left\| \begin{bmatrix} \mathbf{W} \\ \mathbf{E} \end{bmatrix} \right\|_{2,1} \qquad \text{s.t.} \qquad \begin{bmatrix} \mathbf{X}^T & \gamma \mathbf{I} \end{bmatrix} \begin{bmatrix} \mathbf{W} \\ \mathbf{E} \end{bmatrix} = \mathbf{Y}, \qquad (13)$$

where $\mathbf{I} \in \mathbb{R}^{n \times n}$ is an identity matrix. Denote $m = n + d$. Let $\mathbf{A} = \begin{bmatrix} \mathbf{X}^T & \gamma \mathbf{I} \end{bmatrix} \in \mathbb{R}^{n \times m}$ and $\mathbf{U} = \begin{bmatrix} \mathbf{W} \\ \mathbf{E} \end{bmatrix} \in \mathbb{R}^{m \times c}$, then the problem in Eq. (13) can be written as:

$$\min_{\mathbf{U}} \left\| \mathbf{U} \right\|_{2,1} \qquad \text{s.t.} \qquad \mathbf{A}\mathbf{U} = \mathbf{Y} \qquad (14)$$

This optimization problem Eq. (14) has been widely used in the Multiple Measurement Vector (MMV) model in signal processing community. It was generally felt that the $\ell_{2,1}$-norm minimization problem is much more difficult to solve than the $\ell_1$-norm minimization problem. Existing algorithms usually reformulate it as a second-order cone programming (SOCP) or semidefinite programming (SDP) problem, which can be solved by interior point method or the bundle method. However, solving SOCP or SDP is computationally very expensive, which limits their use in practice. Recently, an efficient algorithm was proposed to solve the specific problem Eq. (14) by complicatedly reformulating the problem as a min-max problem and then applying the proximal method to solve it [25]. The reported results show that the algorithm is more efficient than existing algorithms. However, the algorithm is a gradient descent type method and converges very slow. Moreover, the algorithm is derived to solve the specific problem, and can not be applied directly to solve other general $\ell_{2,1}$-norm minimization problem.

In the next subsection, we will propose a very simple but at the same time much more efficient method to solve this problem. Theoretical analysis guarantees that the proposed method will converge to the global optimum. More importantly, this method is very easy to implement and can be readily used to solve other general $\ell_{2,1}$-norm minimization problem.

### 4.2 An Efficient Algorithm to Solve the Constrained Problem

The Lagrangian function of the problem in Eq. (14) is

$$\mathcal{L}(\mathbf{U}) = \left\| \mathbf{U} \right\|_{2,1} - Tr(\mathbf{\Lambda}^T (\mathbf{A}\mathbf{U} - \mathbf{Y})). \qquad (15)$$

Taking the derivative of $\mathcal{L}(\mathbf{U})$ w.r.t $\mathbf{U}$, and setting the derivative to zero, we have:

$$\frac{\partial \mathcal{L}(\mathbf{U})}{\partial \mathbf{U}} = 2\mathbf{D}\mathbf{U} - \mathbf{A}^T\mathbf{\Lambda} = \mathbf{0}, \tag{16}$$

where $\mathbf{D}$ is a diagonal matrix with the $i$-th diagonal element as[1]

$$d_{ii} = \frac{1}{2\left\|\mathbf{u}^i\right\|_2}. \tag{17}$$

Left multiplying the two sides of Eq. (16) by $\mathbf{A}\mathbf{D}^{-1}$, and using the constraint $\mathbf{A}\mathbf{U} = \mathbf{Y}$, we have:

$$2\mathbf{A}\mathbf{U} - \mathbf{A}\mathbf{D}^{-1}\mathbf{A}^T\mathbf{\Lambda} = \mathbf{0}$$
$$\Rightarrow 2\mathbf{Y} - \mathbf{A}\mathbf{D}^{-1}\mathbf{A}^T\mathbf{\Lambda} = \mathbf{0}$$
$$\Rightarrow \mathbf{\Lambda} = 2(\mathbf{A}\mathbf{D}^{-1}\mathbf{A}^T)^{-1}\mathbf{Y} \tag{18}$$

Substitute Eq. (18) into Eq. (16), we arrive at:

$$\mathbf{U} = \mathbf{D}^{-1}\mathbf{A}^T(\mathbf{A}\mathbf{D}^{-1}\mathbf{A}^T)^{-1}\mathbf{Y}. \tag{19}$$

Since the problem in Eq. (14) is a convex problem, $\mathbf{U}$ is a global optimum solution to the problem if and only if the Eq. (19) is satisfied. Note that $\mathbf{D}$ is dependent to $\mathbf{U}$ and thus is also a unknown variable. We propose an iterative algorithm in this paper to obtain the solution $\mathbf{U}$ such that Eq. (19) is satisfied, and prove in the next subsection that the proposed iterative algorithm will converge to the global optimum.

The algorithm is described in Algorithm 1. In each iteration, $\mathbf{U}$ is calculated with the current $\mathbf{D}$, and then $\mathbf{D}$ is updated based on the current calculated $\mathbf{U}$. The iteration procedure is repeated until the algorithm converges.

---

**Data**: $\mathbf{A} \in \mathbb{R}^{n \times m}$, $\mathbf{Y} \in \mathbb{R}^{n \times c}$
**Result**: $\mathbf{U} \in \mathbb{R}^{m \times c}$
Set $t = 0$. Initialize $\mathbf{D}_t \in \mathbb{R}^{m \times m}$ as an identity matrix
**repeat**
    Calculate $\mathbf{U}_{t+1} = \mathbf{D}_t^{-1}\mathbf{A}^T(\mathbf{A}\mathbf{D}_t^{-1}\mathbf{A}^T)^{-1}\mathbf{Y}$.
    Calculate the diagonal matrix $\mathbf{D}_{t+1}$, where the $i$-th diagonal element is $\frac{1}{2\left\|\mathbf{u}_{t+1}^i\right\|_2}$.
    $t = t + 1$.
**until** *Converges*

---

**Algorithm 1:** An efficient iterative algorithm to solve the optimization problem in Eq. (14).

## 4.3 Algorithm Analysis

The Algorithm 1 monotonically decreases the objective of the problem in Eq. (14) in each iteration. To prove it, we need the following lemma:

**Lemma 1.** *For any nonzero vectors $\mathbf{u}, \mathbf{u}_t \in \mathbb{R}^c$, the following inequality holds:*

$$\left\|\mathbf{u}\right\|_2 - \frac{\left\|\mathbf{u}\right\|_2^2}{2\left\|\mathbf{u}_t\right\|_2} \leq \left\|\mathbf{u}_t\right\|_2 - \frac{\left\|\mathbf{u}_t\right\|_2^2}{2\left\|\mathbf{u}_t\right\|_2}. \tag{20}$$

*Proof.* Beginning with an obvious inequality $(\sqrt{v} - \sqrt{v_t})^2 \geq 0$, we have

$$(\sqrt{v} - \sqrt{v_t})^2 \geq 0 \Rightarrow v - 2\sqrt{vv_t} + v_t \geq 0 \Rightarrow \sqrt{v} - \frac{v}{2\sqrt{v_t}} \leq \frac{\sqrt{v_t}}{2} \Rightarrow \sqrt{v} - \frac{v}{2\sqrt{v_t}} \leq \sqrt{v_t} - \frac{v_t}{2\sqrt{v_t}} \tag{21}$$

Substitute the $v$ and $v_t$ in Eq. (21) by $\left\|\mathbf{u}\right\|_2^2$ and $\left\|\mathbf{u}_t\right\|_2^2$ respectively, we arrive at the Eq. (20). $\qquad\square$

The convergence of the Algorithm 1 is summarized in the following theorem:

**Theorem 1.** *The Algorithm 1 will monotonically decrease the objective of the problem in Eq. (14) in each iteration, and converge to the global optimum of the problem.*

*Proof.* It can easily verified that Eq. (19) is the solution to the following problem:

$$\min_{\mathbf{U}} Tr(\mathbf{U}^T \mathbf{D} \mathbf{U}) \qquad \text{s.t.} \qquad \mathbf{A}\mathbf{U} = \mathbf{Y} \qquad (22)$$

Thus in the $t$ iteration,

$$\mathbf{U}_{t+1} = \arg \min_{\mathbf{U} \; \mathbf{A}\mathbf{U}=\mathbf{Y}} Tr\mathbf{U}^T \mathbf{D}_t \mathbf{U}, \qquad (23)$$

which indicates that

$$Tr(\mathbf{U}_{t+1}^T \mathbf{D}_t \mathbf{U}_{t+1}) \leq Tr(\mathbf{U}_t^T \mathbf{D}_t \mathbf{U}_t). \qquad (24)$$

That is to say,

$$\sum_{i=1}^{m} \frac{\left\| \mathbf{u}_{t+1}^i \right\|_2^2}{2 \left\| \mathbf{u}_t^i \right\|_2} \leq \sum_{i=1}^{m} \frac{\left\| \mathbf{u}_t^i \right\|_2^2}{2 \left\| \mathbf{u}_t^i \right\|_2}, \qquad (25)$$

where vectors $\mathbf{u}_t^i$ and $\mathbf{u}_{t+1}^i$ denote the $i$-th row of matrices $\mathbf{U}_t$ and $\mathbf{U}_{t+1}$, respectively.

On the other hand, according to Lemma 1, for each $i$ we have

$$\left\| \mathbf{u}_{t+1}^i \right\|_2 - \frac{\left\| \mathbf{u}_{t+1}^i \right\|_2^2}{2 \left\| \mathbf{u}_t^i \right\|_2} \leq \left\| \mathbf{u}_t^i \right\|_2 - \frac{\left\| \mathbf{u}_t^i \right\|_2^2}{2 \left\| \mathbf{u}_t^i \right\|_2}. \qquad (26)$$

Thus the following inequality holds:

$$\sum_{i=1}^{m} \left( \left\| \mathbf{u}_{t+1}^i \right\|_2 - \frac{\left\| \mathbf{u}_{t+1}^i \right\|_2^2}{2 \left\| \mathbf{u}_t^i \right\|_2} \right) \leq \sum_{i=1}^{m} \left( \left\| \mathbf{u}_t^i \right\|_2 - \frac{\left\| \mathbf{u}_t^i \right\|_2^2}{2 \left\| \mathbf{u}_t^i \right\|_2} \right). \qquad (27)$$

Combining Eq. (25) and Eq. (27), we arrive at

$$\sum_{i=1}^{m} \left\| \mathbf{u}_{t+1}^i \right\|_2 \leq \sum_{i=1}^{m} \left\| \mathbf{u}_t^i \right\|_2. \qquad (28)$$

That is to say,

$$\left\| \mathbf{U}_{t+1} \right\|_{2,1} \leq \left\| \mathbf{U}_t \right\|_{2,1}. \qquad (29)$$

Thus the Algorithm 1 will monotonically decrease the objective of the problem in Eq. (14) in each iteration $t$. In the convergence, $\mathbf{U}_t$ and $\mathbf{D}_t$ will satisfy the Eq. (19). As the problem in Eq. (14) is a convex problem, satisfying the Eq. (19) indicates that $\mathbf{U}$ is a global optimum solution to the problem in Eq. (14). Therefore, the Algorithm 1 will converge to the global optimum of the problem (14). □

Note that in each iteration, the Eq. (19) can be solved efficiently. First, $\mathbf{D}$ is a diagonal matrix and thus $\mathbf{D}^{-1}$ is also diagonal with the $i$-th diagonal element as $d_{ii}^{-1} = 2 \left\| \mathbf{u}^i \right\|_2$. Second, the term $\mathbf{Z} = (\mathbf{A}\mathbf{D}^{-1}\mathbf{A}^T)^{-1}\mathbf{Y}$ in Eq. (19) can be efficiently obtained by solving the linear equation:

$$(\mathbf{A}\mathbf{D}^{-1}\mathbf{A}^T)\mathbf{Z} = \mathbf{Y}. \qquad (30)$$

Empirical results show that the convergence is fast and only a few iterations are needed to converge. Therefore, the proposed method can be applied to large scale problem in practice.

It is worth to point out that the proposed method can be easily extended to solve other $\ell_{2,1}$-norm minimization problem. For example, considering a general $\ell_{2,1}$-norm minimization problem as follows:

$$\min_{\mathbf{U}} f(\mathbf{U}) + \sum_k \left\| \mathbf{A}_k \mathbf{U} + \mathbf{B}_k \right\|_{2,1} \qquad \text{s.t.} \qquad \mathbf{U} \in \mathcal{C} \qquad (31)$$

The problem can be solved by solve the following problem iteratively:

$$\min_{\mathbf{U}} f(\mathbf{U}) + \sum_k Tr((\mathbf{A}_k \mathbf{U} + \mathbf{B}_k)^T \mathbf{D}_k (\mathbf{A}_k \mathbf{U} + \mathbf{B}_k)) \qquad \text{s.t.} \qquad \mathbf{U} \in \mathcal{C} \qquad (32)$$

where $\mathbf{D}_k$ is a diagonal matrix with the $i$-th diagonal element as $\frac{1}{2\|(\mathbf{A}_k \mathbf{U}+\mathbf{B}_k)^i\|_2}$. Similar theoretical analysis can be used to prove that the iterative method will converge to a local minimum. If the problem Eq. (31) is a convex problem, *i.e.*, $f(\mathbf{U})$ is a convex function and $\mathcal{C}$ is a convex set, then the iterative method will converge to the global minimum.

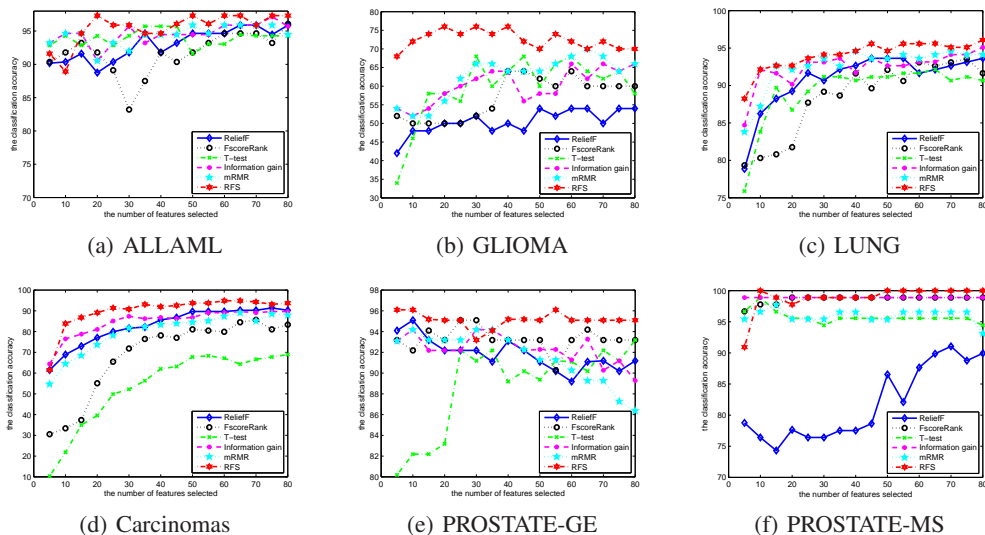

<div align="center">(a) ALLAML      (b) GLIOMA      (c) LUNG</div>

<div align="center">(d) Carcinomas      (e) PROSTATE-GE      (f) PROSTATE-MS</div>

Figure 1: Classification accuracy comparisons of six feature selection algorithms on 6 data sets. SVM with 5-fold cross validation is used for classification. RFS is our method.

## 5 Experimental Results

In order to validate the performance of our feature selection method, we applied our method into two bioinformatics applications, gene expression and mass spectrometry classifications. In our experiments, we used five publicly available microarray data sets and one Mass Spectrometry (MS) data sets: ALLAML data set [6], the malignant glioma (GLIOMA) data set [17], the human lung carcinomas (LUNG) data set [2], Human Carcinomas (Carcinomas) data set [24, 27], Prostate Cancer gene expression (Prostate-GE) data set [23] for microarray data; and Prostate Cancer (Prostate-MS) [20] for MS data. The Support Vector Machine (SVM) classifier is employed to these data sets, using 5-fold cross-validation.

### 5.1 Data Sets Descriptions

We give a brief description on all data sets used in our experiments as follows.

**ALLAML** data set contains in total 72 samples in two classes, ALL and AML, which contain 47 and 25 samples, respectively. Every sample contains 7,129 gene expression values.

**GLIOMA** data set contains in total 50 samples in four classes, cancer glioblastomas (CG), non-cancer glioblastomas (NG), cancer oligodendrogliomas (CO) and non-cancer oligodendrogliomas (NO), which have 14, 14, 7,15 samples, respectively. Each sample has 12625 genes. Genes with minimal variations across the samples were removed. For this data set, intensity thresholds were set at 20 and 16,000 units. Genes whose expression levels varied $< 100$ units between samples, or varied $< 3$ fold between any two samples, were excluded. After preprocessing, we obtained a data set with 50 samples and 4433 genes.

**LUNG** data set contains in total 203 samples in five classes, which have 139, 21, 20, 6,17 samples, respectively. Each sample has 12600 genes. The genes with standard deviations smaller than 50 expression units were removed and we obtained a data set with 203 samples and 3312 genes.

**Carcinomas** data set composed of total 174 samples in eleven classes, prostate, bladder/ureter, breast, colorectal, gastroesophagus, kidney, liver, ovary, pancreas, lung adenocarcinomas, and lung squamous cell carcinoma, which have 26, 8, 26, 23, 12, 11, 7, 27, 6, 14, 14 samples, respectively. In the original data [24], each sample contains 12533 genes. In the preprocessed data set [27], there are 174 samples and 9182 genes.

Table 1: Classification Accuracy of SVM using 5-fold cross validation. Six feature selection methods are compared. RF: ReliefF, F-s: F-score, IG: Information Gain, and RFS: our method.

| | Average accuracy of top 20 features (%) | | | | | | Average accuracy of top 80 features (%) | | | | | |
|---|---|---|---|---|---|---|---|---|---|---|---|---|
| | RF | F-s | T-test | IG | mRMR | RFS | RF | F-s | T-test | IG | mRMR | RFS |
| ALLAML | 90.36 | 89.11 | 92.86 | 93.21 | 93.21 | **95.89** | 95.89 | 96.07 | 94.29 | 95.71 | 94.46 | **97.32** |
| GLIOMA | 50 | 50 | 56 | 60 | 62 | **74** | 54 | 60 | 58 | 66 | 66 | **70** |
| LUNG | 91.68 | 87.7 | 89.22 | 93.1 | 92.61 | **93.63** | 93.63 | 91.63 | 90.66 | 95.1 | 94.12 | **96.07** |
| Carcinom. | 79.88 | 65.48 | 49.9 | 85.09 | 78.22 | **91.38** | 90.24 | 83.33 | 68.91 | 89.65 | 87.92 | **93.66** |
| Pro-GE | 92.18 | 95.09 | 92.18 | 92.18 | 93.18 | **95.09** | 91.18 | 93.18 | 93.18 | 89.27 | 86.36 | **95.09** |
| Pro-MS | 76.41 | **98.89** | 95.56 | **98.89** | 95.42 | **98.89** | 89.93 | 98.89 | 94.44 | 98.89 | 93.14 | **100** |
| Average | 80.09 | 81.04 | 79.29 | 87.09 | 85.78 | **91.48** | 85.81 | 87.18 | 83.25 | 89.10 | 87 | **92.02** |

**Prostate-GE** data set has in total 102 samples in two classes tumor and normal, which have 52 and 50 samples, respectively. The original data set contains 12600 genes. In our experiment, intensity thresholds were set at 100 C16000 units. Then we filtered out the genes with max/min $\leq 5$ or (max-min) $\leq 50$. After preprocessing, we obtained a data set with 102 samples and 5966 genes.

**Prostate-MS** data can be obtained from the FDA-NCI Clinical Proteomics Program Databank [20]. This MS data set consists of 190 samples diagnosed as benign prostate hyperplasia, 63 samples considered as no evidence of disease, and 69 samples diagnosed as prostate cancer. The samples diagnosed as benign prostate hyperplasia as well as samples having no evidence of prostate cancer were pooled into one set making 253 control samples, whereas the other 69 samples are the cancer samples.

## 5.2 Classification Accuracy Comparisons

All data sets are standardized to be zero-mean and normalized by standard deviation. SVM classifier has been individually performed on all data sets using 5-fold cross-validation. We utilize the linear kernel with the parameter $C = 1$. We compare our feature selection method (called as RFS) to several popularly used feature selection methods in bioinformatics, such as $F$-statistic [4], reliefF [11, 13], mRMR [19], t-test, and information gain [21]. Because the above data sets are for multi-class classification problem, we don't compare to $\ell_1$-SVM, HHSVM and other methods that were designed for binary classification.

Fig. 1 shows the classification accuracy comparisons of all five feature selection methods on six data sets. Table 1 shows the detailed experimental results using SVM. We compute the average accuracy using the top 20 and top 80 features for all feature selection approaches. Obviously our approaches outperform other methods significantly. With top 20 features, our method is around 5%-12% better than other methods all six data sets.

## 6 Conclusions

In this paper, we proposed a new efficient and robust feature selection method with emphasizing joint $\ell_{2,1}$-norm minimization on both loss function and regularization. The $\ell_{2,1}$-norm based regression loss function is robust to outliers in data points and also efficient in calculation. Motivated by previous work, the $\ell_{2,1}$-norm regularization is used to select features across all data points with joint sparsity. We provided an efficient algorithm with proved convergence. Our method has been applied into both genomic and proteomic biomarkers discovery. Extensive empirical studies have been performed on two bioinformatics tasks, six data sets, to demonstrate the performance of our method.

## 7 Acknowledgements

This research was funded by US NSF-CCF-0830780, 0939187, 0917274, NSF DMS-0915228, NSF CNS-0923494, 1035913.

## Footnotes

[1]When $\mathbf{u}^i = 0$, then $d_{ii} = 0$ is a subgradient of $\left\|\mathbf{U}\right\|_{2,1}$ w.r.t. $\mathbf{u}^i$. However, we can not set $d_{ii} = 0$ when $\mathbf{u}^i = 0$, otherwise the derived algorithm can not be guaranteed to converge. Two methods can be used to solve this problem. First, we will see from Eq.(19) that we only need to calculate $\mathbf{D}^{-1}$, so we can let the $i$-th element of $\mathbf{D}^{-1}$ as $2\left\|\mathbf{u}^i\right\|_2$. Second, we can regularize $d_{ii}$ as $d_{ii} = \frac{1}{2\sqrt{(\mathbf{u}^i)^T\mathbf{u}^i+\varsigma}}$, and the derived algorithm can be proved to minimize the regularized $\ell_{2,1}$-norms of $\mathbf{U}$ (defined as $\sum_{i=1}^{n}\sqrt{(\mathbf{u}^i)^T\mathbf{u}^i+\varsigma}$) instead of the $\ell_{2,1}$-norms of $\mathbf{U}$. It is easy to see that the regularized $\ell_{2,1}$-norms of $\mathbf{U}$ approximates the $\ell_{2,1}$-norms of $\mathbf{U}$ when $\varsigma \to 0$.

# References

[1] A. Argyriou, T. Evgeniou, and M. Pontil. Multi-task feature learning. *NIPS*, pages 41–48, 2007.

[2] A. Bhattacharjee, W. G. Richards, and et. al. Classification of human lung carcinomas by mRNA expression profiling reveals distinct adenocarcinoma subclasses. *Proceedings of the National Academy of Sciences*, 98(24):13790–13795, 2001.

[3] P. Bradley and O. Mangasarian. Feature selection via concave minimization and support vector machines. *ICML*, 1998.

[4] C. Ding and H. Peng. Minimum redundancy feature selection from microarray gene expression data. *Proceedings of the Computational Systems Bioinformatics*, 2003.

[5] C. Ding, D. Zhou, X. He, and H. Zha. R1-PCA: Rotational invariant L1-norm principal component analysis for robust subspace factorization. *Proc. Int'l Conf. Machine Learning (ICML)*, June 2006.

[6] S. P. Fodor. DNA SEQUENCING: Massively Parallel Genomics. *Science*, 277(5324):393–395, 1997.

[7] I. Guyon and A. Elisseeff. An introduction to variable and feature selection. *J. Machine Learning Research*, 2003.

[8] I. Guyon, J.Weston, S. Barnhill, and V. Vapnik. Gene selection for cancer classification using support vector machines. *Machine Learning*, 46(1):389, 2002.

[9] M. A. Hall and L. A. Smith. Feature selection for machine learning: Comparing a correlation-based filter approach to the wrapper. 1999.

[10] H. Huang and C. Ding. Robust tensor factorization using r1 norm. *CVPR 2008*, pages 1–8, 2008.

[11] K. Kira and L. A. Rendell. A practical approach to feature selection. In *A Practical Approach to Feature Selection*, pages 249–256, 1992.

[12] R. Kohavi and G. H. John. Wrappers for feature subset selection. *Artificial Intelligence*, 97(1-2):273–324, 1997.

[13] I. Kononenko. Estimating attributes: Analysis and extensions of RELIEF. In *European Conference on Machine Learning*, pages 171–182, 1994.

[14] P. Langley. Selection of relevant features in machine learning. In *AAAI Fall Symposium on Relevance*, pages 140–144, 1994.

[15] H. Liu and H. Motoda. *Feature Selection for Knowledge Discovery and Data Mining*. Springer, 1998.

[16] D. Luo, C. Ding, and H. Huang. Towards structural sparsity: An explicit $\ell_2/\ell_0$ approach. *ICDM*, 2010.

[17] C. L. Nutt, D. R. Mani, R. A. Betensky, P. Tamayo, J. G. Cairncross, C. Ladd, U. Pohl, C. Hartmann, and M. E. Mclaughlin. Gene expression-based classification of malignant gliomas correlates better with survival than histological classification. *Cancer Res.*, 63:1602–1607, 2003.

[18] G. Obozinski, B. Taskar, and M. Jordan. Multi-task feature selection. *Technical report, Department of Statistics, University of California, Berkeley*, 2006.

[19] H. Peng, F. Long, and C. Ding. Feature selection based on mutual information: Criteria of max-dependency, max-relevance, and min-redundancy. *IEEE Trans. Pattern Analysis and Machine Intelligence*, 27, 2005.

[20] P. C. Petricoin EF, Ornstein DK. Serum proteomic patterns for detection of prostate cancer. *J Natl Cancer Inst.*, 94(20):1576–8, 2002.

[21] L. E. Raileanu and K. Stoffel. Theoretical comparison between the gini index and information gain criteria. *Univeristy of Neuchatel*, 2000.

[22] Y. Saeys, I. Inza, and P. Larranaga. A review of feature selection techniques in bioinformatics. *Bioinformatics*, 23(19):2507–2517, 2007.

[23] D. Singh, P. Febbo, K. Ross, and et al. Gene expression correlates of clinical prostate cancer behavior. *Cancer Cell*, pages 203–209, 2002.

[24] A. I. Su, J. B. Welsh, L. M. Sapinoso, and et al. Molecular classification of human carcinomas by use of gene expression signatures. *Cancer Research*, 61:7388–7393, 2001.

[25] L. Sun, J. Liu, J. Chen, and J. Ye. Efficient recovery of jointly sparse vectors. In *Neural Information Processing Systems*, 2009.

[26] L. Wang, J. Zhu, and H. Zou. Hybrid huberized support vector machines for microarray classification. *ICML*, 2007.

[27] K. Yang, Z. Cai, J. Li, and G. Lin. A stable gene selection in microarray data analysis. *BMC Bioinformatics*, 7:228, 2006.

[28] M. Yuan and Y. Lin. Model selection and estimation in regression with grouped variables. *Journal of the Royal Statistical Society: Series B*, 68:49–67, 2005.

